# Categories and Functional Units: An Infinite Hierarchical Model for Brain Activations

**Danial Lashkari**      **Ramesh Sridharan**      **Polina Golland**
Computer Science and Artificial Intelligence Laboratory
Massachusetts Institute of Technology
Cambridge, MA 02139
`{danial, rameshvs, polina}@csail.mit.edu`

## Abstract

We present a model that describes the structure in the responses of different brain areas to a set of stimuli in terms of *stimulus categories* (clusters of stimuli) and *functional units* (clusters of voxels). We assume that voxels within a unit respond similarly to all stimuli from the same category, and design a nonparametric hierarchical model to capture inter-subject variability among the units. The model explicitly encodes the relationship between brain activations and fMRI time courses. A variational inference algorithm derived based on the model learns categories, units, and a set of unit-category activation probabilities from data. When applied to data from an fMRI study of object recognition, the method finds meaningful and consistent clusterings of stimuli into categories and voxels into units.

## 1  Introduction

The advent of functional neuroimaging techniques, in particular fMRI, has for the first time provided non-invasive, large-scale observations of brain processes. Functional imaging techniques allow us to directly investigate the high-level functional organization of the human brain. Functional specificity is a key aspect of this organization and can be studied along two separate dimensions: 1) which sets of stimuli or cognitive tasks are treated similarly by the brain, and 2) which areas of the brain have similar functional properties. For instance, in the studies of visual object recognition the first question defines object categories intrinsic to the visual system, while the second characterizes regions with distinct profiles of selectivity. To answer these questions, fMRI studies examine the responses of all relevant brain areas to as many stimuli as possible within the domain under study. Novel methods of analysis are needed to extract the patterns of functional specificity from the resulting high-dimensional data.

Clustering is a natural choice for answering questions we pose here regarding functional specificity with respect to both stimuli and voxels. Applying clustering in the space of stimuli identifies stimuli that induce similar patterns of response and has been recently used to discover object *categories* from responses in the human inferior temporal cortex [1]. Applying clustering in the space of brain locations seeks voxels that show similar functional responses [2, 3, 4, 5]. We will refer to a cluster of voxels with similar responses as a *functional unit*.

In this paper, we present a model to investigate the interactions between these two aspects of functional specificity. We make the natural assumptions that functional units are organized based on their responses to the categories of stimuli and the categories of stimuli can be characterized by the responses they induce in the units. Therefore, categories and units are interrelated and informative about each other. Our generative model simultaneously learns the specificity structure in the space of both stimuli and voxels. We use a block co-clustering framework to model the relationship between clusters of stimuli and brain locations [6]. In order to account for variability across subjects in a group study, we assume a hierarchical model where a group-level structure generates the clustering of voxels in different subjects (Fig. 1). A nonparametric prior enables the model to search the space

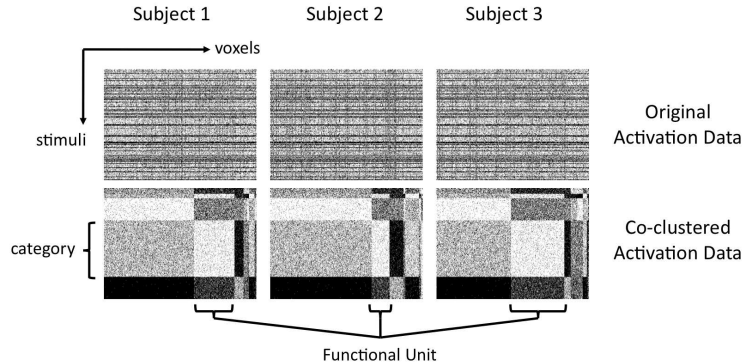

Figure 1: Co-clustering fMRI data across subjects. The first row shows a hypothetical data set of brain activations. The second row shows the same data after co-clustering, where rows and columns are re-ordered based on the membership in categories and functional units.

of different numbers of clusters. Furthermore, we tailor the method specifically to brain imaging by including a model of fMRI signals [7]. Most prior work applies existing machine learning algorithms to functional neuroimaging data. In contrast, our Bayesian integration of the co-clustering model with the model of fMRI signals informs each level of the model about the uncertainties of inference in the other levels. As a result, the algorithm is better suited to handling the high levels of noise in fMRI observations.

We apply our method to a group fMRI study of visual object recognition where 8 subjects are presented with 69 distinct images. The algorithm finds a clustering of the set of images into a number of categories along with a clustering of voxels in different subjects into units. We find that the learned categories and functional units are indeed meaningful and consistent.

**Related Work**  Different variants of co-clustering algorithms have found applications in biological data analysis [8, 9, 10]. Our model is closely related to the probabilistic formulations of co-clustering [11, 12] and the application of Infinite Relational Models to co-clustering [13]. Prior work in the applications of advanced machine learning techniques to fMRI has mainly focused on supervised learning, which requires prior knowledge of stimulus categories [14]. Unsupervised learning methods such as Independent Component Analysis (ICA) have also been applied to fMRI data to decompose it into a set of spatial and temporal (functional) components [15, 16]. ICA assumes an additive model for the data and allows spatially overlapping components. However, neither of these assumptions is appropriate for studying functional specificity. For instance, an fMRI response that is a weighted combination of a component selective for category A and another component selective for category B may be better described by selectivity for a new category (the union of both). We also note that Formal Concept Analysis, which is closely related to the idea of block co-clustering, has been recently applied to neural data from visual studies in monkeys [17].

## 2  Model

Our model consists of three main components:

    I. Co-clustering structure expressing the relationship between the clustering of stimuli (categories) and the clustering of brain voxels (functional units),

    II. Hierarchical structure expressing the variability among functional units across subjects,

    III. Signal model expressing the relationship between voxel activations and observed fMRI time courses.

The co-clustering level is the key element of the model that encodes the interactions between stimulus categories and functional units. Due to the differences in the level of noise among subjects, we do not expect to find the same set of functional units in all subjects. We employ the structure of the Hierarchical Dirichlet Processes (HDP) [18] to account for this fact. The first two components of the model jointly explain how different brain voxels are activated by each stimulus in the experiment. The third component of the model links these binary activations to the observed fMRI time courses

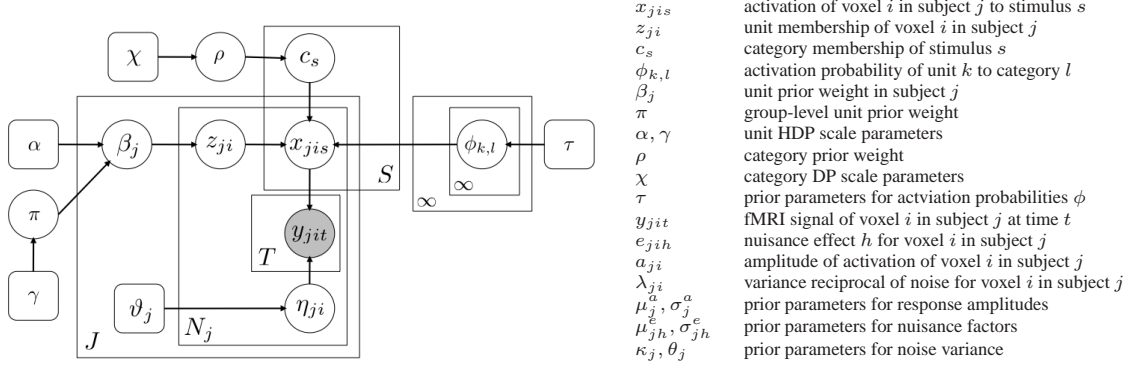

| | |
|---|---|
| $x_{jis}$ | activation of voxel $i$ in subject $j$ to stimulus $s$ |
| $z_{ji}$ | unit membership of voxel $i$ in subject $j$ |
| $c_s$ | category membership of stimulus $s$ |
| $\phi_{k,l}$ | activation probability of unit $k$ to category $l$ |
| $\beta_j$ | unit prior weight in subject $j$ |
| $\pi$ | group-level unit prior weight |
| $\alpha, \gamma$ | unit HDP scale parameters |
| $\rho$ | category prior weight |
| $\chi$ | category DP scale parameters |
| $\tau$ | prior parameters for actviation probabilities $\phi$ |
| $y_{jit}$ | fMRI signal of voxel $i$ in subject $j$ at time $t$ |
| $e_{jih}$ | nuisance effect $h$ for voxel $i$ in subject $j$ |
| $a_{ji}$ | amplitude of activation of voxel $i$ in subject $j$ |
| $\lambda_{ji}$ | variance reciprocal of noise for voxel $i$ in subject $j$ |
| $\mu_j^a, \sigma_j^a$ | prior parameters for response amplitudes |
| $\mu_{jh}^e, \sigma_{jh}^e$ | prior parameters for nuisance factors |
| $\kappa_j, \theta_j$ | prior parameters for noise variance |

Figure 2: The graphical representation of our model where the set of voxel response variables $(a_{ji}, e_{jih}, \lambda_{ji})$ and their corresponding prior parameters $(\mu_j^a, \sigma_j^a, \mu_h^e, \sigma_h^e, \kappa_j, \theta_j)$ are denoted by $\eta_{ji}$ and $\vartheta_j$, respectively.

of voxels. Sec. 2.1 presents the hierarchical co-clustering part of the model that includes both the first and the second components above. Sec. 2.2 presents the fMRI signal model that integrates the estimation of voxel activations with the rest of the model. Sec. 2.3 outlines the variational algorithm that we employ for inference. Fig. 2 shows the graphical model for the joint distribution of the variables in the model.

## 2.1 Nonparametric Hierarchical Co-clustering Model

Let $x_{jis} \in \{0, 1\}$ be an activation variable that indicates whether stimulus $s$ activates voxel $i$ in subject $j$. The co-clustering model describes the distribution of voxel activations $x_{jis}$ based on the category and the functional units to which stimulus $s$ and voxel $i$ belong. We assume that all voxels within functional unit $k$ have the same probability $\phi_{k,l}$ of being activated by a particular category $l$ of stimuli. Let $\boldsymbol{z} = \{z_{ji}\}, (z_{ji} \in \{1, 2, \cdots\})$ be the set of unit memberships of voxels and $\boldsymbol{c} = \{c_s\}, (c_s \in \{1, 2, \cdots\})$ the set of category memberships of the stimuli. Our model of co-clustering assumes:

$$x_{jis} \mid z_{ji}, c_s, \boldsymbol{\phi} \stackrel{i.i.d.}{\sim} \text{Bernoulli}(\phi_{z_{ji}, c_s}). \tag{1}$$

The set $\boldsymbol{\phi} = \{\phi_{k,l}\}$ of the probabilities of activation of functional units to different categories summarizes the structure in the responses of voxels to stimuli.

We use the stick-breaking formulation of HDP [18] to construct an infinite hierarchical prior for voxel unit memberships:

$$z_{ji} \mid \beta_j \stackrel{i.i.d.}{\sim} \text{Mult}(\beta_j), \tag{2}$$

$$\beta_j \mid \pi \stackrel{i.i.d.}{\sim} \text{Dir}(\alpha\pi), \tag{3}$$

$$\pi \mid \gamma \sim \text{GEM}(\gamma). \tag{4}$$

Here, $\text{GEM}(\gamma)$ is a distribution over infinitely long vectors $\pi = [\pi_1, \pi_2, \cdots]^T$, named after Griffiths, Engen and McCloskey [19]. This distribution is defined as:

$$\pi_k = v_k \prod_{k'=1}^{k-1} (1 - v_{k'}), \qquad v_k \mid \gamma \stackrel{i.i.d.}{\sim} \text{Beta}(1, \gamma), \tag{5}$$

where the components of the generated vectors $\pi$ sum to one with probability 1. In subject $j$, voxel memberships are distributed according to subject-specific weights of functional units $\beta_j$. The weights $\beta_j$ are in turn generated by a Dirichlet distribution centered around $\pi$ with a degree of variability determined by $\alpha$. Therefore, $\pi$ acts as the group-level expected value of the subject-specific weights. With this prior over the unit memberships of voxels $\boldsymbol{z}$, the model in principle allows an infinite number of functional units; however, for any finite set of voxels, a finite number of units is sufficient to include all voxels.

We do not impose a similar hierarchical structure on the clustering of stimuli among subjects. Conceptually, we assume that stimulus categories reflect how the human brain has evolved to

organize the processing of stimuli within a system and are therefore identical across subjects. Even if any variability exists, it will be hard to learn such a complex structure from data since we can present relatively few stimuli in each experiment. Hence, we assume identical clustering $c$ in the space of stimuli for all subjects, with a Dirichlet process prior:

$$c_s \mid \rho \overset{i.i.d.}{\sim} \mathrm{Mult}(\rho),$$

$$\rho \mid \chi \ \sim \ \mathrm{GEM}(\chi). \tag{6}$$

Finally, we construct the prior distribution for unit-category activation probabilities $\phi$:

$$\phi_{k,l} \overset{i.i.d.}{\sim} \mathrm{Beta}(\tau_1, \tau_2). \tag{7}$$

## 2.2 Model of fMRI Signals

Functional MRI yields a noisy measure of average neuronal activation in each brain voxel at different time points. The standard linear time-invariant model of fMRI signals expresses the contribution of each stimulus by the convolution of the spike train of stimulus onsets with a hemodynamic response function (HRF) [20]. The HRF peaks at about 6-9 seconds, modeling an intrinsic delay between the underlying neural activity and the measured fMRI signal. Accordingly, measured signal $y_{jit}$ in voxel $i$ of subject $j$ at time $t$ is modeled as:

$$y_{jit} = \sum_s b_{jis} G_{st} + \sum_h e_{jih} F_{ht} + \epsilon_{jit}, \tag{8}$$

where $G_{st}$ is the model regressor for stimulus $s$, $F_{ht}$ represents nuisance factor $h$, such as a baseline or a linear temporal trend, at time $t$ and $\epsilon_{jit}$ is gaussian noise. We use the simplifying assumption throughout that $\epsilon_{jit} \overset{i.i.d.}{\sim} \mathrm{Normal}(0, \lambda_{ji}^{-1})$. In the absence of any priors, the response $b_{jis}$ of voxel $i$ to stimulus $s$ can be estimated by solving the least squares regression problem.

Unfortunately, fMRI signal does not have a meaningful scale and may vary greatly across trials and experiments. In order to use this data for inferences about brain function across subjects, sessions, and stimuli, we need to transform it into a standard and meaningful space. The binary activation variables $x$, introduced in the previous section, achieve this transformation by assuming that in response to any stimulus a voxel is either in an active or a non-active state, similar to [7]. If voxel $i$ is activated by stimulus $s$, i.e., if $x_{jis} = 1$, its response takes positive value $a_{ji}$ that specifies the voxel-specific amplitude of response; otherwise, its response remains 0. We can write $b_{jis} = a_{ji} x_{jis}$ and assume that $a_{ji}$ represents uninteresting variability in fMRI signal. When making inference on binary activation variable $x_{jis}$, we consider not only the response, but also the level of noise and responses to other stimuli. Therefore, the binary activation variables can be directly compared across different subjects, sessions, and experiments.

We assume the following priors on voxel response variables:

$$e_{jih} \ \sim \ \mathrm{Normal}\left(\mu_{jh}^e, \sigma_{jh}^e\right), \tag{9}$$

$$a_{ji} \ \sim \ \mathrm{Normal}_+\left(\mu_j^a, \sigma_j^a\right), \tag{10}$$

$$\lambda_{ji} \ \sim \ \mathrm{Gamma}\left(\kappa_j, \theta_j\right), \tag{11}$$

where $\mathrm{Normal}_+$ defines a normal distribution constrained to only take positive values.

## 2.3 Algorithm

The size of common fMRI data sets and the space of hidden variables in our model makes stochastic inference methods, such as Gibbs sampling, prohibitively slow. Currently, there is no faster split-merge-type sampling technique that can be applied to hierarchical nonparametric models [18]. We therefore choose a variational Bayesian inference scheme, which is known to yield faster algorithms.

To formulate the inference for the hierarchical unit memberships, we closely follow the derivation of the Collapsed Variational HDP approximation [21]. We integrate over the subject-specific unit weights $\boldsymbol{\beta} = \{\beta_j\}$ and introduce a set of auxiliary variables $\boldsymbol{r} = \{r_{jk}\}$ that represent the number of tables corresponding to unit (dish) $k$ in subject (restaurant) $j$ according to the Chinese restaurant franchise formulation of HDP [18]. Let $\boldsymbol{h} = \{\boldsymbol{x}, \boldsymbol{z}, \boldsymbol{c}, \boldsymbol{r}, \boldsymbol{a}, \boldsymbol{\phi}, \boldsymbol{e}, \boldsymbol{\lambda}, v, u\}$ denote the set of all unobserved variables. Here, $v = \{v_k\}$ and $u = \{u_l\}$ are the stick breaking fractions corresponding

to distributions $\pi$ and $\rho$, respectively. We approximate the posterior distribution on the hidden variables given the observed data $p(\boldsymbol{h}|\boldsymbol{y})$ by a factorizable distribution $q(\boldsymbol{h})$. The variational method minimizes the Gibbs free energy function $\mathcal{F}[q] = E[\log q(\boldsymbol{h})] - E[\log p(\boldsymbol{y}, \boldsymbol{h})]$ where $E[\cdot]$ indicates expected value with respect to distribution $q$. We assume a distribution $q$ of the form:

$$q(\boldsymbol{h}) = q(\boldsymbol{r}|\boldsymbol{z}) \prod_k q(v_k) \prod_l q(u_l) \prod_{k,l} q(\phi_{k,l}) \prod_s q(c_s) \cdot \prod_{j,i} \left[ q(a_{ji})q(\lambda_{ji})q(z_{ji}) \prod_s q(x_{jis}) \prod_h q(e_{jih}) \right].$$

We apply coordinate descent in the space of $q(\cdot)$ to minimize the free energy. Since we explicitly account for the dependency of the auxiliary variables on unit memberships in the posterior, we can derive closed form update rules for all hidden variables. Due to space constraints in this paper, we present the update rules and their derivations in the Supplementary Material.

Iterative application of the update rules leads to a local minimum of the Gibbs free energy. Since variational solutions are known to be biased toward their initial configurations, the initialization phase becomes critical to the quality of the results. For initialization of the activation variables $x_{jis}$, we estimate $b_{jis}$ in Eq. (8) using least squares regression and for each voxel normalize the estimates to values between $0$ and $1$ using the voxel-wise maximum and minimum. We use the estimates of $\boldsymbol{b}$ to also initialize $\boldsymbol{\lambda}$ and $\boldsymbol{e}$. For memberships, we initialize $q(\boldsymbol{z})$ by introducing the voxels one by one in random order to the collapsed Gibbs sampling scheme [18] constructed for our model with each stimulus as a separate category and the initial $\boldsymbol{x}$ assumed known. We initialize category memberships $\boldsymbol{c}$ by clustering the voxel responses across all subjects. Finally, we set the hyperparameters of the fMRI model such that they match the corresponding statistics computed by least squares regression on the data.

## 3   Results

We demonstrate the performance of the model and the inference algorithm on both synthetic and real data. As a baseline algorithm for comparison, we use the Block Average Co-clustering (BAC) algorithm [6] with the Euclidean distance. First, we show that the hierarchical structure of our algorithm enables us to retrieve the cluster membership more accurately in synthetic group data. Then, we present the results of our method in an fMRI study of visual object recognition.

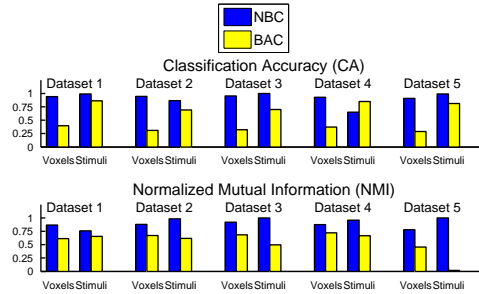

Figure 3: Comparison between our nonparametric Bayesian co-clustering algorithm (NBC) and Block Average Co-clustering (BAC) on synthetic data. Both classicication accuracy (CA) and noramlized mutual information (NMI) are reported.

### 3.1   Synthetic Data

We generate synthetic data from a stochastic process defined by our model with the set of parameters $\gamma = 3$, $\alpha = 100$, $\chi = 1$, and $\tau_1 = \tau_2 = 1$, $N_j = 1000$ voxels, $S = 100$ stimuli, and $J = 4$ subjects. For the model of the fMRI signals, we use parameters that are representative of our experimental setup and the corresponding hyperparameters estimated from the data. We generate 5 data sets with these parameters; they have between 5 to 7 categories and 13 to 21 units. We apply our algorithm directly to time courses in 5 different data sets generated using the above scheme. To apply BAC to the same data sets, we need to first turn the time-courses into voxel-stimulus data. We use the least squares estimates of voxel responses ($b_{jis}$) normalized in the same way as we initialize our fMRI model. We run each algorithm 20 times with different initializations. The BAC algorithm is initialized by the result of a soft $k$-means clustering in the space of voxels. Our method is initialized as explained in the previous section. For BAC, we use the *true* number of clusters while our algorithm is always initialized with 15 clusters.

We evaluate the results of clustering with respect to both voxels and stimuli by comparing clustering results with the ground truth. Since there is no consensus on the best way to compare different clusterings of the same set, here we employ two different clustering distance measures. Let $P(k, k')$ denote the fraction of data points (voxels or stimuli) assigned to cluster $k$ in the ground truth and $k'$

in the estimated clustering. The first measure is the so-called *classification accuracy* (CA), which is defined as the fraction of data points correctly assigned to the true clusters [22]. To compute this measure, we need to first match the cluster indices in our results with the true clustering. We find a one-to-one matching between the two sets of clusters by solving a bipartite graph matching problem. We define the graph such that the two sets of cluster indices represent the nodes and $P(k, k')$ represents the weight of the edge between node $k$ and $k'$. As the second measure, we use the *normalized mutual information* (NMI), which expresses the proportion of the entropy (information) of the ground truth clustering that is shared with the estimated clustering. We define two random variables $X$ and $Y$ that take values in the spaces of the true and the estimated cluster indices, respectively. Assuming a joint distribution $P(X{=}k, Y{=}k') = P(k, k')$, we set $NMI = I(X; Y)/H(X)$. Both measures take values between 0 and 1, with 1 corresponding to perfect clustering.

Fig. 3 presents the clustering quality measures for the two algorithms on the 5 generated data sets. As expected, our method performs consistently better in finding the true clustering structure on data generated by the co-clustering process. Since the two algorithms share the same block co-clustering structure, the advantage of our method is in its model for the hierarchical structure and fMRI signals.

## 3.2 Experiment

We apply our method to data from an fMRI study where 8 subjects view 69 distinct images. Each image is repeated on average about 40 times in one of the two sessions in the experiment. The data includes 42 slices of 1.65mm thickness with in plane voxel size of 1.5mm, aligned with the temporal lobe (ventral visual pathway). As part of the standard preprocessing stream, the data was first motion-corrected separately for the two sessions [23], and then spatially smoothed with a Gaussian kernel of 3mm width. The time course data included 120 volumes per run and from 24 to 40 runs for each subject. We registered the data from the two sessions to the subject's native anatomical space [24]. We removed noisy voxels from the analysis by performing an ANOVA test and only keeping the voxels for which the stimulus regressors significantly explained the variation in the time course (threshold $p{=}10^{-4}$ uncorrected). This procedure selects on average about 6,000 voxels for each subject. Finally, to remove the idiosyncratic aspects of responses in different subjects, such as attention to particular stimuli, we regressed out the subject-average time course from voxel signals after removing the baseline and linear trend. We split trials of each image into two groups of equal size and consider each group as an independent stimulus forming a total of 138 stimuli. Hence, we can examine the consistency of our stimulus categorization with respect to identical trials.

We use $\alpha = 100, \gamma = 5, \chi = 0.1$, and $\tau_1 = \tau_2 = 1$ for the nonparametric prior. We initialize our algorithm 20 times and choose the solution that achieves the lowest Gibbs free energy. Fig. 4 shows the categories that the algorithm finds on the data from all 8 subjects. First, we note that stimulus pairs corresponding to the same image are generally assigned to the same category, confirming the consistency of the resuls across trials. Category 1 corresponds to the scene images and, interestingly, also includes all images of trees. This may suggest a high level category structure that is not merely driven by low level features. Such a structure is even more evident in the 4th category where images of a tiger that has a large face join human faces. Some other animals are clustered together with human bodies in categories 2 and 9. Shoes and cars, which have similar shapes, are clustered together in category 3 while tools are mainly found in category 6.

The interaction between the learned categories and the functional units is summarized in the posterior unit-category activation probabilities $E[\phi_{k,l}]$ ( Fig. 4, right ). The algorithm finds 18 units across all subjects. The largest unit does not show preference for any of the categories. Functional unit 2 is the most selective one and shows high activation for category 4 (faces). This finding agrees with previous studies that have discovered face-selective areas in the brain [25]. Other units show selectivity for different combinations of categories. For instance, Unit 6 prefers categories that mostly include body parts and animals, unit 8 prefers category 1 (scenes and trees), while the selectivity of unit 5 seems to be correlated with the pixel-size of the image.

Our method further learns sets of variables $\{q(z_{ji}{=}k)\}_{i=1}^{N_j}$ that represent the probabilities that different voxels in subject $j$ belong to functional unit $k$. Although the algorithm does not use any information about the spatial location of voxels, we can visualize the posterior membership probabilities in each subject as a spatial map. To see whether there is any degree of spatial consistency in the locations of the learned units across subjects, we align the brains of all subjects with the Montreal

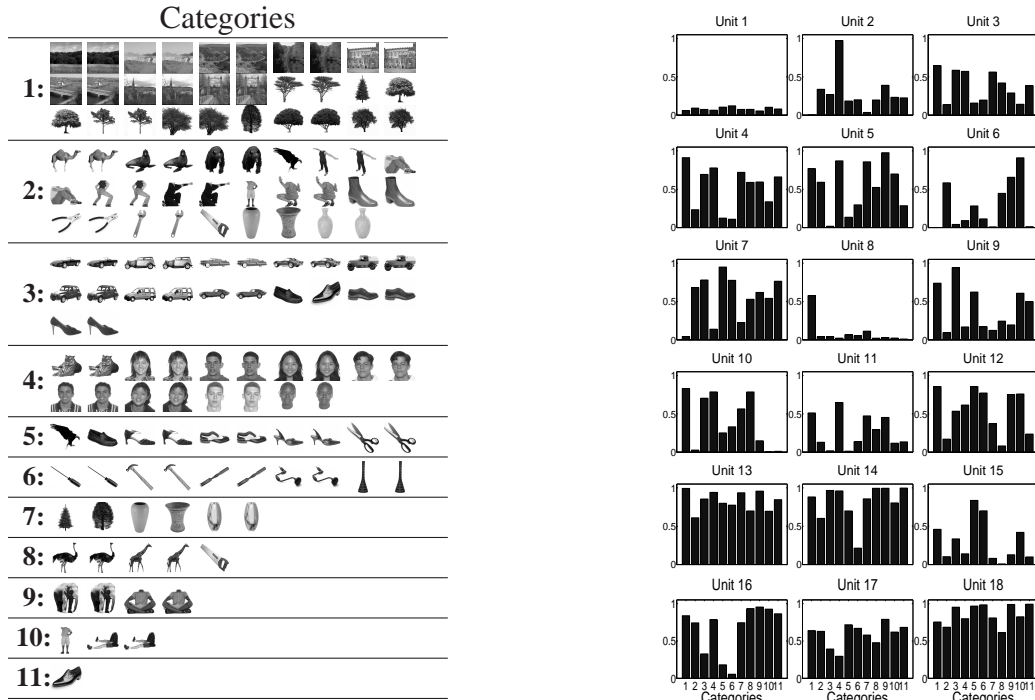

Figure 4: Categories (left) and activation probabilities of functional units ($E[\phi_{k,l}]$) (right) estimated by the algorithm from all 8 subjects in the study.

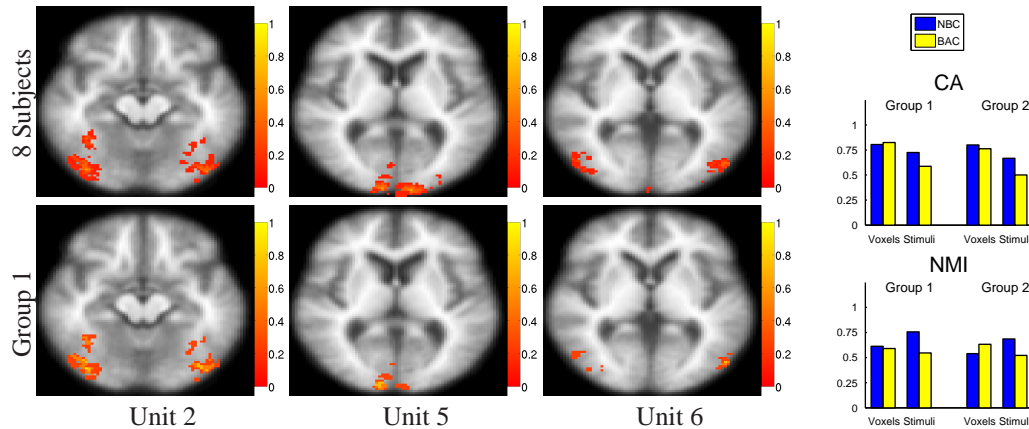

Figure 5: (Left) Spatial maps of functional unit overlap across subjects in the normalized space. For each voxel, we show the fraction of subjects in the group for which the voxel was assigned to the corresponding functional unit. We see that functional units with similar profiles between the two datasets show similar spatial extent as well. (Right) Comparison between the clustering robustness in the results of our algorithm (NBC) and the best results of Block Average Co-clustering (BAC) on the real data.

Neurological Institute coordinate space using affine registration [26]. Fig. 5 (left) shows the average maps across subjects for units 2, 5, and 6 in the normalized space. Despite the relative sparsity of the maps, they have significant overlap across subjects.

As with many other real world applications of clustering, the validation of results is challenging in the absence of ground truth. In order to assess the reliability of the results, we examine their consistency across subjects. We split the 8 subjects into two groups of 4 and perform the analysis on the two group data separately. Fig. 6 (left) shows the categories found for one of the two groups (group 1), which show good agreement with the categories found in the data from all subjects (categories are indexed based on the result of graph matching). As a way to quantify the stability of clustering across subjects, we compute the measures CA and NMI for the results in the two groups

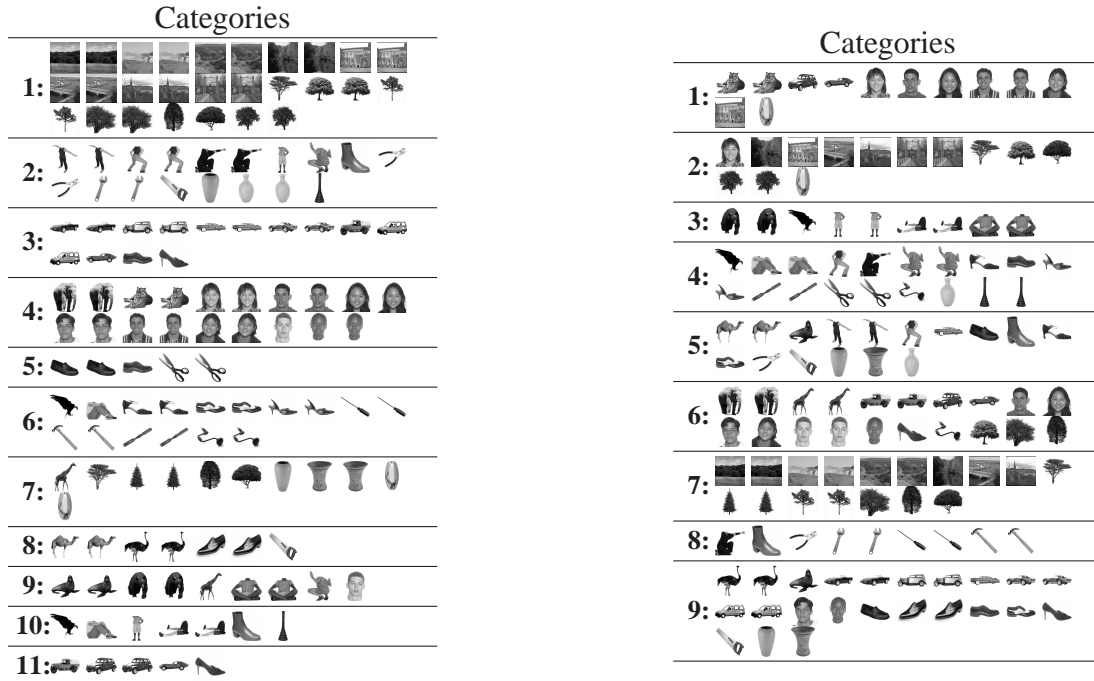

Figure 6: Categories found by our algorithm in group 1 (left) and by BAC in all subjects for $(l, k) = (14, 14)$ (right).

relative to the results in the 8 subjects. We also apply the BAC algorithm to response values estimated via least squares regression in all 8 subjects and the two groups. Since the number of units and categories is not known a priori, we perform the BAC algorithm for all pairs of $(l, k)$ such that $5 \leq l \leq 15$ and $k \in \{10, 12, 14, 16, 18, 20\}$. Fig. 5 (right) compares the clustering measures for our method with those found by the best BAC results in terms of average CA and NMI measures (achieved with $(l, k) = (6, 14)$ for CA, and $(l, k) = (14, 14)$ for NMI). Fig. 6 (right) shows the categories for $(l, k) = (14, 14)$, which appear to lack some of the structures found in our results. We also obtain better measures of stability compared to the best BAC results for clustering stimuli, while the measures are similar for clustering voxels. We note that in contrast to the results of BAC, our first unit is always considerably larger than all the others including about 70% of voxels. This seems neuroscientifically plausible since we expect large areas of the visual cortex to be involved in processing low level features and therefore incapable of distinguishing different objects.

## 4   Conclusion

This paper proposes a model for learning large-scale functional structures in the brain responses of a group of subjects. We assume that the structure can be summarized in terms of functional units with similar responses to categories of stimuli. We derive a variational Bayesian inference scheme for our hierarchical nonparametric Bayesian model and apply it to both synthetic and real data. In an fMRI study of visual object recognition, our method finds meaningful structures in both object categories and functional units.

This work is a step toward devising models for functional brain imaging data that explicitly encode our hypotheses about the structure in the brain functional organization. The assumption that functional units, categories, and their interactions are sufficient to describe the structure, although proved successful here, may be too restrictive in general. A more detailed characterization may be achieved through a feature-based representation where a stimulus can simultaneously be part of several categories (features). Likewise, a more careful treatment of the structure in the organization of brain areas may require incorporating spatial information. In this paper, we show that we can turn such basic insights into principled models that allow us to investigate the structures of interest in a data-driven fashion. By incorporating the properties of brain imaging signals into the model, we better utilize the data for making relevant inferences across subjects.

**Acknowledgments**

We thank Ed Vul, Po-Jang Hsieh, and Nancy Kanwisher for the insight they have offered us throughout our collaboration, and also for providing the fMRI data. This research was supported in part by the NSF grants IIS/CRCNS 0904625, CAREER 0642971, the MIT McGovern Institute Neurotechnology Program grant, and NIH grants NIBIB NAMIC U54-EB005149 and NCRR NAC P41-RR13218.

# References

[1] N. Kriegeskorte, M. Mur, D.A. Ruff, R. Kiani, J. Bodurka, H. Esteky, K. Tanaka, and P.A. Bandettini. Matching categorical object representations in inferior temporal cortex of man and monkey. *Neuron*, 60(6):1126–1141, 2008.

[2] B. Thirion and O. Faugeras. Feature characterization in fMRI data: the Information Bottleneck approach. *MedIA*, 8(4):403–419, 2004.

[3] D. Lashkari and P. Golland. Exploratory fMRI analysis without spatial normalization. In *IPMI*, 2009.

[4] D. Lashkari, E. Vul, N. Kanwisher, and P. Golland. Discovering structure in the space of fMRI selectivity profiles. *NeuroImage*, 50(3):1085–1098, 2010.

[5] D. Lashkari, R. Sridharan, E. Vul, P.J. Hsieh, N. Kanwisher, and P. Golland. Nonparametric hierarchical Bayesian model for functional brain parcellation. In *MMBIA*, 2010.

[6] A. Banerjee, I. Dhillon, J. Ghosh, S. Merugu, and D.S. Modha. A generalized maximum entropy approach to bregman co-clustering and matrix approximation. *JMLR*, 8:1919–1986, 2007.

[7] S. Makni, P. Ciuciu, J. Idier, and J.-B. Poline. Joint detection-estimation of brain activity in functional MRI: a multichannel deconvolution solution. *TSP*, 53(9):3488–3502, 2005.

[8] Y. Cheng and G.M. Church. Biclustering of expression data. In *ISMB*, 2000.

[9] S.C. Madeira and A.L. Oliveira. Biclustering algorithms for biological data analysis: a survey. *TCBB*, 1(1):24–45, 2004.

[10] Y. Kluger, R. Basri, J.T. Chang, and M. Gerstein. Spectral biclustering of microarray data: coclustering genes and conditions. *Genome Research*, 13(4):703–716, 2003.

[11] B. Long, Z.M. Zhang, and P.S. Yu. A probabilistic framework for relational clustering. In *ACM SIGKDD*, 2007.

[12] D. Lashkari and P. Golland. Coclustering with generative models. CSAIL Technical Report, 2009.

[13] C. Kemp, J.B. Tenenbaum, T.L. Griffiths, T. Yamada, and N. Ueda. Learning systems of concepts with an infinite relational model. In *AAAI*, 2006.

[14] K.A. Norman, S.M. Polyn, G.J. Detre, and J.V. Haxby. Beyond mind-reading: multi-voxel pattern analysis of fMRI data. *Trends in Cognitive Sciences*, 10(9):424–430, 2006.

[15] C.F. Beckmann and S.M. Smith. Probabilistic independent component analysis for functional magnetic resonance imaging. *TMI*, 23(2):137–152, 2004.

[16] M.J. McKeown, S. Makeig, G.G. Brown, T.P. Jung, S.S. Kindermann, A.J. Bell, and T.J. Sejnowski. Analysis of fMRI data by blind separation into independent spatial components. *Hum Brain Mapp*, 6(3):160–188, 1998.

[17] D. Endres and P. Földiák. Interpreting the neural code with Formal Concept Analysis. In *NIPS*, 2009.

[18] Y.W. Teh, M.I. Jordan, M.J. Beal, and D.M. Blei. Hierarchical dirichlet processes. *JASA*, 101(476):1566–1581, 2006.

[19] J. Pitman. Poisson–Dirichlet and GEM invariant distributions for split-and-merge transformations of an interval partition. *Combinatorics, Prob, Comput*, 11(5):501–514, 2002.

[20] KJ Friston, AP Holmes, KJ Worsley, JP Poline, CD Frith, RSJ Frackowiak, et al. Statistical parametric maps in functional imaging: a general linear approach. *Hum Brain Mapp*, 2(4):189–210, 1994.

[21] Y.W. Teh, K. Kurihara, and M. Welling. Collapsed variational inference for HDP. In *NIPS*, 2008.

[22] M. Meilă and D. Heckerman. An experimental comparison of model-based clustering methods. *Machine Learning*, 42(1):9–29, 2001.

[23] R.W. Cox and A. Jesmanowicz. Real-time 3D image registration for functional MRI. *Magn Reson Med*, 42(6):1014–1018, 1999.

[24] D.N. Greve and B. Fischl. Accurate and robust brain image alignment using boundary-based registration. *NeuroImage*, 48(1):63–72, 2009.

[25] N. Kanwisher and G. Yovel. The fusiform face area: a cortical region specialized for the perception of faces. *R Soc Lond Phil Trans, Series B*, 361(1476):2109–2128, 2006.

[26] J. Talairach and P. Tournoux. *Co-planar Stereotaxic Atlas of the Human Brain*. Thieme, New York, 1988.

